# Learning Sparse Representations of High Dimensional Data on Large Scale Dictionaries

**Zhen James Xiang**    **Hao Xu**    **Peter J. Ramadge**
Department of Electrical Engineering, Princeton University
Princeton, NJ 08544, USA
{zxiang,haoxu,ramadge}@princeton.edu

## Abstract

Learning sparse representations on data adaptive dictionaries is a state-of-the-art method for modeling data. But when the dictionary is large and the data dimension is high, it is a computationally challenging problem. We explore three aspects of the problem. First, we derive new, greatly improved screening tests that quickly identify codewords that are guaranteed to have zero weights. Second, we study the properties of random projections in the context of learning sparse representations. Finally, we develop a hierarchical framework that uses incremental random projections and screening to learn, in small stages, a hierarchically structured dictionary for sparse representations. Empirical results show that our framework can learn informative hierarchical sparse representations more efficiently.

## 1 Introduction

Consider approximating a $p$-dimensional data point $\mathbf{x}$ by a linear combination $\mathbf{x} \approx \mathbf{Bw}$ of $m$ (possibly linearly dependent) codewords in a dictionary $\mathbf{B} = [\mathbf{b}_1, \mathbf{b}_2, \ldots, \mathbf{b}_m]$. Doing so by imposing the additional constraint that $\mathbf{w}$ is a *sparse* vector, i.e., $\mathbf{x}$ is approximated as a weighted sum of only a few codewords in the dictionary, has recently attracted much attention [1]. As a further refinement, when there are many data points $\mathbf{x}_j$, the dictionary $\mathbf{B}$ can be optimized to make the representations $\mathbf{w}_j$ as sparse as possible. This leads to the following problem. Given $n$ data points in $\mathbb{R}^p$ organized as matrix $\mathbf{X} = [\mathbf{x}_1, \mathbf{x}_2, \ldots, \mathbf{x}_n] \in \mathbb{R}^{p \times n}$, we want to learn a dictionary $\mathbf{B} = [\mathbf{b}_1, \mathbf{b}_2, \ldots, \mathbf{b}_m] \in \mathbb{R}^{p \times m}$ and sparse representation weights $\mathbf{W} = [\mathbf{w}_1, \mathbf{w}_2, \ldots, \mathbf{w}_n] \in \mathbb{R}^{m \times n}$ so that each data point $\mathbf{x}_j$ is well approximated by $\mathbf{Bw}_j$ with $\mathbf{w}_j$ a sparse vector:

$$\min_{\mathbf{B},\mathbf{W}} \quad \frac{1}{2}\|\mathbf{X} - \mathbf{BW}\|_F^2 + \lambda\|\mathbf{W}\|_1$$
$$\text{s.t.} \quad \|\mathbf{b}_i\|_2^2 \leq 1, \quad \forall i = 1, 2, \ldots, m. \tag{1}$$

Here $\|\cdot\|_F$ and $\|\cdot\|_1$ denote the Frobenius norm and element-wise $l_1$-norm of a matrix, respectively.

There are two advantages to this representation method. First, the dictionary $\mathbf{B}$ is *adapted to the data*. In the spirit of many modern approaches (e.g. PCA, SMT [2], tree-induced bases [3,4]), rather than fixing $\mathbf{B}$ *a priori* (e.g. Fourier, wavelet, DCT), problem (1) assumes minimal prior knowledge and uses sparsity as a cue to learn a dictionary adapted to the data. Second, the new representation $\mathbf{w}$ is obtained by a *nonlinear* mapping of $\mathbf{x}$. Algorithms such as Laplacian eigenmaps [5] and LLE [6], also use nonlinear mappings $\mathbf{x} \mapsto \mathbf{w}$. By comparison, $l_1$-regularization enjoys a simple formulation with a single tuning parameter ($\lambda$). In many other approaches (including [2–4]), although the codewords in $\mathbf{B}$ are cleverly chosen, the new representation $\mathbf{w}$ is simply a linear mapping of $\mathbf{x}$, e.g. $\mathbf{w} = \mathbf{B}^\dagger \mathbf{x}$. In this case, training a linear model on $\mathbf{w}$ cannot learn nonlinear structure in the data. As a final point, we note that the human visual cortex uses similar mechanisms to encode visual scenes [7] and sparse representation has exhibited superior performance on difficult computer vision problems such as face [8] and object [9] recognition.

The challenge, however, is that solving the non-convex optimization problem(1) is computationally expensive. Most state-of-the-art algorithms solve (1) by iteratively optimizing $\mathbf{W}$ and $\mathbf{B}$. For a fixed $\mathbf{B}$, optimizing $\mathbf{W}$ requires solving $n$, $p$-dimensional, lasso problems of size $m$. Using LARS [10] with a Cholesky-based implementation, each lasso problem has a computation cost of $O(mp\kappa + m\kappa^2)$, where $\kappa$ is the number of nonzero coefficients [11]. For a fixed $\mathbf{W}$, optimizing $\mathbf{B}$ is a least squares problem of $pm$ variables and $m$ constraints. In an efficient algorithm [12], the dual formulation has only $m$ variables but still requires inverting $m \times m$ matrices ($O(m^3)$ complexity).

To address this challenge, we examine decomposing a large dictionary learning problem into a set of smaller problems. First (§2), we explore dictionary screening [13, 14], to select a subset of codewords to use in each Lasso optimization. We derive two new screening tests that are *significantly better* than existing tests when the data points and codewords are highly correlated, a typical scenario in sparse representation applications [15]. We also provide simple geometric intuition for guiding the derivation of screening tests. Second (§3), we examine projecting data onto a lower dimensional space so that we can control information flow in our hierarchical framework and solve sparse representations with smaller $p$. We identify an important property of the data that's implicitly assumed in sparse representation problems (scale indifference) and study how random projection preserves this property. These results are inspired by [16] and related work in compressed sensing. Finally (§4), we develop a framework for learning a hierarchical dictionary (similar in spirit to [17] and DBN [18]). To do so we exploit our results on screening and random projection and impose a zero-tree like structured sparsity constraint on the representation. This constraint is similar to the formulation in [19]. The key difference is that we learn the sparse representation stage-wise in layers and use the exact zero-tree sparsity constraint to utilize the information in previous layers to simplify the computation, whereas [19] uses a convex relaxation to approximate the structured sparsity constraint and learns the sparse representation (of all layers) by solving a single large optimization problem. Our idea of using incremental random projections is inspired by the work in [20, 21]. Finally, unlike [12] (that addresses the same computational challenge), we focus on a high level reorganization of the computations rather than improving basic optimization algorithms. Our framework can be combined with all existing optimization algorithms, e.g. [12], to attain faster results.

## 2 Reducing the Dictionary By Screening

In this section we assume that all data points and codewords are normalized: $\|\mathbf{x}_j\|_2 = \|\mathbf{b}_i\|_2 = 1, 1 \le j \le n, 1 \le i \le m$ (we discuss the implications of this assumption in §3). When $\mathbf{B}$ is fixed, finding the optimal $\mathbf{W}$ in (1) requires solving $n$ subproblems. The $j^{th}$ subproblem finds $\mathbf{w}_j$ for $\mathbf{x}_j$. For notational simplicity, in this section we drop the index $j$ and denote $\mathbf{x} = \mathbf{x}_j, \mathbf{w} = \mathbf{w}_j = [w_1, w_2, \ldots, w_m]^T$. Each subproblem is then of the form:

$$\min_{w_1, w_2, \ldots, w_m} \quad \frac{1}{2}\|\mathbf{x} - \sum_{i=1}^{m} w_i \mathbf{b}_i\|_2^2 + \lambda \sum_{i=1}^{m} |w_i|. \tag{2}$$

To address the challenge of solving (2) for large $m$, we first explore simple screening tests that identify and discard codewords $\mathbf{b}_i$ guaranteed to have optimal solution $\tilde{w}_i = 0$. El Ghaoui's SAFE rule [13] is an example of a simple screening test. We introduce some simple geometric intuition for screening and use this to derive new tests that are significantly better than existing tests for the type of problems of interest here. To this end, it will help to consider the dual problem of (2):

$$\max_{\boldsymbol{\theta}} \quad \frac{1}{2}\|\mathbf{x}\|_2^2 - \frac{\lambda^2}{2}\|\boldsymbol{\theta} - \frac{\mathbf{x}}{\lambda}\|_2^2 \tag{3}$$
$$\text{s.t.} \quad |\boldsymbol{\theta}^T \mathbf{b}_i| \le 1 \quad \forall i = 1, 2, \ldots, m.$$

As is well known (see the supplemental material), the optimal solution of the primal problem $\tilde{\mathbf{w}} = [\tilde{w}_1, \tilde{w}_2, \ldots, \tilde{w}_m]^T$ and the optimal solution of the dual problem $\tilde{\boldsymbol{\theta}}$ are related through:

$$\mathbf{x} = \sum_{i=1}^{m} \tilde{w}_i \mathbf{b}_i + \lambda \tilde{\boldsymbol{\theta}}, \qquad \tilde{\boldsymbol{\theta}}^T \mathbf{b}_i \in \begin{cases} \{\text{sign } \tilde{w}_i\} & \text{if } \tilde{w}_i \ne 0, \\ [-1, 1] & \text{if } \tilde{w}_i = 0. \end{cases} \tag{4}$$

The dual formulation gives useful geometric intuition. Since $\|\mathbf{x}\|_2 = \|\mathbf{b}_i\|_2 = 1$, $\mathbf{x}$ and all $\mathbf{b}_i$ lie on the unit sphere $S^{p-1}$ (Fig.1(a)). For $\mathbf{y}$ on $S^{p-1}$, $P(\mathbf{y}) = \{\mathbf{z} : \mathbf{z}^T \mathbf{y} = 1\}$ is the tangent hyperplane

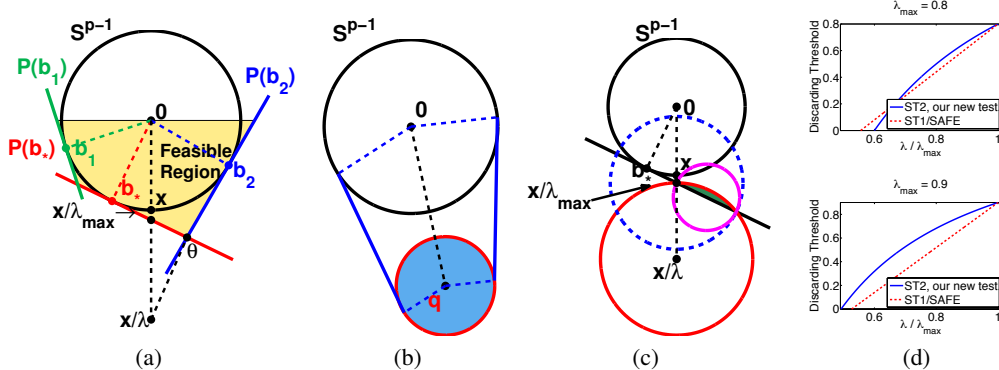

Figure 1: **(a)** Geometry of the dual problem. **(b)** Illustration of a sphere test. **(c)** The solid red, dotted blue and solid magenta circles leading to sphere tests ST1/SAFE, ST2, ST3, respectively. **(d)** The thresholds in ST2 and ST1/SAFE when $\lambda_{\max} = 0.8$ (top) and $\lambda_{\max} = 0.9$ (bottom). A higher threshold yields a better test.

of $S^{p-1}$ at $\mathbf{y}$ and $H(\mathbf{y}) = \{\mathbf{z} : \mathbf{z}^T\mathbf{y} \leq 1\}$ is the corresponding closed half space containing the origin. The constraints in (3) indicate that feasible $\boldsymbol{\theta}$ must be in $H(\mathbf{b}_i)$ and $H(-\mathbf{b}_i)$ for all $i$. To find $\tilde{\boldsymbol{\theta}}$ that maximizes the objective in (3), we must find a feasible $\boldsymbol{\theta}$ closest to $\mathbf{x}/\lambda$. By (4), if $\tilde{\boldsymbol{\theta}}$ is not on $P(\mathbf{b}_i)$ or $P(-\mathbf{b}_i)$, then $\tilde{w}_i = 0$ and we can safely discard $\mathbf{b}_i$ from problem (2).

Let $\lambda_{\max} = \max_i |\mathbf{x}^T\mathbf{b}_i|$ and $\mathbf{b}_* \in \{\pm\mathbf{b}_i\}_{i=1}^m$ be selected so that $\lambda_{\max} = \mathbf{x}^T\mathbf{b}_*$. Note that $\boldsymbol{\theta} = \mathbf{x}/\lambda_{\max}$ is a feasible solution for (3). $\lambda_{\max}$ is also the largest $\lambda$ for which (2) has a nonzero solution. If $\lambda > \lambda_{\max}$, then $\mathbf{x}/\lambda$ itself is feasible, making it the optimal solution. Since it is not on any hyperplane $P(\mathbf{b}_i)$ or $P(-\mathbf{b}_i)$, $\tilde{w}_i = 0$, $i = 1, \ldots, m$. Hence we assume that $\lambda \leq \lambda_{\max}$.

These observations can be used for screening as follows. If we know that $\tilde{\boldsymbol{\theta}}$ is within a region $\mathcal{R}$, then we can discard those $\mathbf{b}_i$ for which the tangent hyperplanes $P(\mathbf{b}_i)$ and $P(-\mathbf{b}_i)$ don't intersect $\mathcal{R}$, since by (4) the corresponding $\tilde{w}_i$ will be 0. Moreover, if the region $\mathcal{R}$ is contained in a closed ball (e.g. the shaded blue area in Fig.1(b)) centered at $\mathbf{q}$ with radius $r$, i.e., $\{\boldsymbol{\theta}: \|\boldsymbol{\theta} - \mathbf{q}\|_2 \leq r\}$, then one can discard all $\mathbf{b}_i$ for which $|\mathbf{q}^T\mathbf{b}_i|$ is smaller than a threshold determined by the common tangent hyperplanes of the spheres $\|\boldsymbol{\theta} - \mathbf{q}\|_2 = r$ and $S^{p-1}$. This "sphere test" is made precise in the following lemma (All lemmata are proved in the supplemental material).

**Lemma 1.** *If the solution $\tilde{\boldsymbol{\theta}}$ of (3) satisfies $\|\tilde{\boldsymbol{\theta}} - \mathbf{q}\|_2 \leq r$, then $|\mathbf{q}^T\mathbf{b}_i| < (1 - r) \Rightarrow \tilde{w}_i = 0$.*

El Ghaoui's SAFE rule [13] is a sphere test of the simplest form. To see this, note that $\mathbf{x}/\lambda_{\max}$ is a feasible point of (3), so the optimal $\boldsymbol{\theta}$ cannot be further away from $\mathbf{x}/\lambda$ than $\mathbf{x}/\lambda_{\max}$. Therefore we have the constraint : $\|\tilde{\boldsymbol{\theta}} - \mathbf{x}/\lambda\|_2 \leq 1/\lambda - 1/\lambda_{\max}$ (solid red ball in Fig.1(c)). Plugging in $\mathbf{q} = \mathbf{x}/\lambda$ and $r = 1/\lambda - 1/\lambda_{\max}$ into Lemma 1 yields El Ghaoui's SAFE rule:

**Sphere Test # 1 (ST1/SAFE):** If $|\mathbf{x}^T\mathbf{b}_i| < \lambda - 1 + \lambda/\lambda_{\max}$, then $\tilde{w}_i = 0$.

Note that the SAFE rule is weakest when $\lambda_{\max}$ is large, i.e., when the codewords are very similar to the data points, a frequent situation in applications [15]. To see that there is room for improvement, consider the constraint: $\boldsymbol{\theta}^T\mathbf{b}_* \leq 1$. This puts $\tilde{\boldsymbol{\theta}}$ in the intersection of the previous closed ball (solid red) and $H(\mathbf{b}_*)$. This is indicated by the shaded green region in Fig. 1(c). Since this intersection is small when $\lambda_{\max}$ is large, a better test results by selecting $\mathcal{R}$ to be the shaded green region. However, to simplify the test, we relax $\mathcal{R}$ to a closed ball and use the sphere test of Lemma 1. Two relaxations, the solid magenta ball and the dotted blue ball in Fig. 1(c), are detailed in the following lemma.

**Lemma 2.** *If $\boldsymbol{\theta}$ satisfies (a) $\|\boldsymbol{\theta} - \mathbf{x}/\lambda\|_2 \leq 1/\lambda - 1/\lambda_{\max}$ and (b) $\boldsymbol{\theta}^T\mathbf{b}_* \leq 1$, then $\boldsymbol{\theta}$ satisfies*

$$\|\boldsymbol{\theta} - (\mathbf{x}/\lambda - (\lambda_{\max}/\lambda - 1)\mathbf{b}_*)\|_2 \leq \sqrt{1/\lambda_{\max}^2 - 1}(\lambda_{\max}/\lambda - 1), \quad and \quad (5)$$

$$\|\boldsymbol{\theta} - \mathbf{x}/\lambda_{\max}\|_2 \leq 2\sqrt{1/\lambda_{\max}^2 - 1}(\lambda_{\max}/\lambda - 1). \quad (6)$$

By Lemma 2, since $\tilde{\boldsymbol{\theta}}$ satisfies (a) and (b), it satisfies (5) and (6). We start with (6) because of its similarity to the closed ball constraint used to derive ST1/SAFE (solid red ball). Plugging $\mathbf{q} = \mathbf{x}/\lambda_{\max}$ and $r = 2\sqrt{1/\lambda_{\max}^2 - 1}(\lambda_{\max}/\lambda - 1)$ into Lemma 1 yields our first new test:

**Sphere Test # 2 (ST2):**     If $|\mathbf{x}^T\mathbf{b}_i| < \lambda_{\max}(1 - 2\sqrt{1/\lambda_{\max}^2 - 1}(\lambda_{\max}/\lambda - 1))$, then $\tilde{w}_i = 0$.

Since ST2 and ST1/SAFE both test $|\mathbf{x}^T\mathbf{b}_i|$ against thresholds, we can compare the tests by plotting their thresholds. We do so for $\lambda_{\max} = 0.8, 0.9$ in Fig.1(d). The thresholds must be positive and large to be useful. ST2 is most useful when $\lambda_{\max}$ is large. Indeed, we have the following lemma:

**Lemma 3.** *When* $\lambda_{\max} > \sqrt{3}/2$*, if ST1/SAFE discards* $\mathbf{b}_i$*, then ST2 also discards* $\mathbf{b}_i$*.*

Finally, to use the closed ball constraint (5), we plug in $\mathbf{q} = \mathbf{x}/\lambda - (\lambda_{\max}/\lambda - 1)\mathbf{b}_*$ and $r = \sqrt{1/\lambda_{\max}^2 - 1}(\lambda_{\max}/\lambda - 1)$ into Lemma 1 to obtain a second new test:

**Sphere Test # 3 (ST3):**
   If $|\mathbf{x}^T\mathbf{b}_i - (\lambda_{\max} - \lambda)\mathbf{b}_*^T\mathbf{b}_i| < \lambda(1 - \sqrt{1/\lambda_{\max}^2 - 1}(\lambda_{\max}/\lambda - 1))$, then $\tilde{w}_i = 0$.

ST3 is slightly more complex. It requires finding $\mathbf{b}_*$ and computing a weighted sum of inner products. But ST3 is always better than ST2 since its sphere lies strictly inside that of ST2:

**Lemma 4.** *Given any* $\mathbf{x}, \mathbf{b}_*$ *and* $\lambda$*, if ST2 discards* $\mathbf{b}_i$*, then ST3 also discards* $\mathbf{b}_i$*.*

To summarize, ST3 completely outperforms ST2, and when $\lambda_{\max}$ is larger than $\sqrt{3}/2 \approx 0.866$, ST2 completely outperforms ST1/SAFE. Empirical comparisons are given in §5.

By making two passes through the dictionary, the above tests can be efficiently implemented on large-scale dictionaries that can't fit in memory. The first pass holds $\mathbf{x}, \mathbf{u}, \mathbf{b}_i \in \mathbb{R}^p$ in memory at once and computes $\mathbf{u}(i) = \mathbf{x}^T\mathbf{b}_i$. By simple bookkeeping, after pass one we have $\mathbf{b}_*$ and $\lambda_{\max}$. The second pass holds $\mathbf{u}, \mathbf{b}_*, \mathbf{b}_i$ in memory at once, computes $\mathbf{b}_*^T\mathbf{b}_i$ and executes the test.

## 3   Random Projections of the Data

In §4 we develop a framework for learning a hierarchical dictionary and this involves the use of random data projections to control information flow to the levels of the hierarchy. The motivation for using random projections will become clear, and is specifically discussed, in §4. Here we lay some groundwork by studying basic properties of random projections in learning sparse representations.

We first revisit the normalization assumption $\|\mathbf{x}_j\|_2 = \|\mathbf{b}_i\|_2 = 1, 1 \leq j \leq n, 1 \leq i \leq m$ in §2. The assumption that all codewords are normalized: $\|\mathbf{b}_i\|_2 = 1$, is necessary for (1) to be meaningful, otherwise we can increase the scale of $\mathbf{b}_i$ and inversely scale the $i^{th}$ row of $\mathbf{W}$ to lower the loss. The assumption that all data points are normalized: $\|\mathbf{x}_j\|_2 = 1$, warrants a more careful examination. To see this, assume that the data $\{\mathbf{x}_j\}_{j=1}^n$ are samples from an underlying low dimensional smooth manifold $\mathcal{X}$ and that one desires a correspondence between codewords and local regions on $\mathcal{X}$. Then we require the following *scale indifference* (SI) property to hold:

**Definition 1.** $\mathcal{X}$ *satisfies the SI property if* $\forall \mathbf{x}_1, \mathbf{x}_2 \in \mathcal{X}$*, with* $\mathbf{x}_1 \neq \mathbf{x}_2$*, and* $\forall \gamma \neq 0$*,* $\mathbf{x}_1 \neq \gamma\mathbf{x}_2$*.*

Intuitively, SI means that $\mathcal{X}$ doesn't contain points differing only in scale and it implies that points $\mathbf{x}_1, \mathbf{x}_2$ from distinct regions on $\mathcal{X}$ will use different codewords in their representation. SI is usually implicitly assumed [9,15] but it will be important for what follows to make the condition explicit. SI is true in many typical applications of sparse representation. For example, for image signals when we are interested in the image content regardless of image luminance. When SI holds we can indeed normalize the data points to $S^{p-1} = \{\mathbf{x} : \|\mathbf{x}\|_2 = 1\}$.

Since a random projection of the original data doesn't preserve the normalization $\|\mathbf{x}_j\|_2 = 1$, it's important for the random projection to preserve the SI property so that it is reasonable to renormalize the projected data. We will show that this is indeed the case under certain assumptions. Suppose we use a random projection matrix $\mathbf{T} \in \mathbb{R}^{d \times p}$, with orthonormal rows, to project the data to $\mathbb{R}^d$ ($d < p$) and use $\mathbf{TX}$ as the new data matrix. Such $\mathbf{T}$ can be generated by running the Gram-Schmidt procedure on $d$, $p$-dimensional random row vectors with i.i.d. Gaussian entries. It's known that for certain sets $\mathcal{X}$, with high probability random projection preserves pairwise distances:

$$(1 - \epsilon)\sqrt{d/p} \leq \frac{\|\mathbf{Tx}_1 - \mathbf{Tx}_2\|_2}{\|\mathbf{x}_1 - \mathbf{x}_2\|_2} \leq (1 + \epsilon)\sqrt{d/p}. \tag{7}$$

For example, when $\mathcal{X}$ contains only $\kappa$-sparse vectors, we only need $d \geq O(\kappa \ln(p/\kappa))$ and when $\mathcal{X}$ is a $K$-dimensional Riemannian submanifold, we only need $d \geq O(K \ln p)$ [16]. We will show that when the pairwise distances are preserved as in (7), the SI property will also be preserved:

**Theorem 1.** *Define $S(\mathcal{X}) = \{\mathbf{z} : \mathbf{z} = \gamma\mathbf{x}, \mathbf{x} \in \mathcal{X}, |\gamma| \leq 1\}$. If $\mathcal{X}$ satisfies SI and $\forall(\mathbf{x}_1, \mathbf{x}_2) \in S(\mathcal{X}) \times S(\mathcal{X})$ (7) is satisfied, then $T(\mathcal{X}) = \{\mathbf{z} : \mathbf{z} = \mathbf{Tx}, \mathbf{x} \in \mathcal{X}\}$ also satisfies SI.*

*Proof.* If $T(\mathcal{X})$ doesn't satisfy SI, then by Definition 1, $\exists(\mathbf{x}_1, \mathbf{x}_2) \in \mathcal{X} \times \mathcal{X}, \gamma \notin \{0, 1\}$ s.t.: $\mathbf{Tx}_1 = \gamma\mathbf{Tx}_2$. Without loss of generality we can assume that $|\gamma| \leq 1$ (otherwise we can exchange the positions of $\mathbf{x}_1$ and $\mathbf{x}_2$). Since $\mathbf{x}_1$ and $\gamma\mathbf{x}_2$ are both in $S(\mathcal{X})$, using (7) gives that $\|\mathbf{x}_1 - \gamma\mathbf{x}_2\|_2 \leq \|\mathbf{Tx}_1 - \gamma\mathbf{Tx}_2\|_2/((1-\epsilon)\sqrt{d/p}) = 0$. So $\mathbf{x}_1 = \gamma\mathbf{x}_2$. This contradicts the SI property of $\mathcal{X}$. □

For example, if $\mathcal{X}$ contains only $\kappa$-sparse vectors, so does $S(\mathcal{X})$. If $\mathcal{X}$ is a Riemannian submanifold, so is $S(\mathcal{X})$. Therefore applying random projections to these $\mathcal{X}$ will preserve SI with high probability. For the case of $\kappa$-sparse vectors, under some strong conditions, we can prove that random projection always preserves SI. (Proofs of the theorems below are in the supplemental material.)

**Theorem 2.** *If $\mathcal{X}$ satisfies SI and has a $\kappa$-sparse representation using dictionary $\mathbf{B}$, then the projected data $T(\mathcal{X})$ satisfies SI if $(2\kappa - 1)M(\mathbf{TB}) < 1$, where $M(\cdot)$ is matrix mutual coherence.*

Combining (7) with Theorem 1 or 2 provides an important insight: the projected data $\mathbf{TX}$ contains rough information about the original data $\mathcal{X}$ and we can continue to use the formulation (1) on $\mathbf{TX}$ to extract such information. Actually, if we do this for a Riemannian submanifold $\mathcal{X}$, then we have:

**Theorem 3.** *Let the data points lie on a $K$-dimensional compact Riemannian submanifold $\mathcal{X} \subset \mathbb{R}^p$ with volume $V$, conditional number $1/\tau$, and geodesic covering regularity $R$ (see [16]). Assume that in the optimal solution of (1) for the projected data (replacing $\mathbf{X}$ with $\mathbf{TX}$), data points $\mathbf{Tx}_1$ and $\mathbf{Tx}_2$ have nonzero weights on the same set of $\kappa$ codewords. Let $\mathbf{w}_j$ be the new representation of $\mathbf{x}_j$ and $\mu_i = \|\mathbf{Tx}_j - \mathbf{Bw}_j\|_2$ be the length of the residual ($j = 1, 2$). With probability $1 - \rho$:*

$$\begin{aligned}
\|\mathbf{x}_1 - \mathbf{x}_2\|_2^2 &\leq (p/d)(1+\epsilon_1)(1+\epsilon_2)(\|\mathbf{w}_1 - \mathbf{w}_2\|_2^2 + 2\mu_1^2 + 2\mu_2^2) \\
\|\mathbf{x}_1 - \mathbf{x}_2\|_2^2 &\geq (p/d)(1-\epsilon_1)(1-\epsilon_2)(\|\mathbf{w}_1 - \mathbf{w}_2\|_2^2),
\end{aligned} \tag{8}$$

*with $\epsilon_1 = O((\frac{K \ln(NVR\tau^{-1})\ln(1/\rho)}{d})^{0.5-\eta})$ (for any small $\eta > 0$) and $\epsilon_2 = (\kappa - 1)M(\mathbf{B})$.*

Therefore the distances between the sparse representation weights reflect the original data point distances. We believe a similar result should also hold when $\mathcal{X}$ contains only $\kappa$-sparse vectors.

## 4  Learning a Hierarchical Dictionary

Our hierarchical framework decomposes a large dictionary learning problem into a sequence of smaller hierarchically structured dictionary learning problems. The result is a tree of dictionaries. High levels of the tree give course representations, deeper levels give more detailed representations, and the codewords at the leaves form the final dictionary. The tree is grown top-down in $l$ levels by refining the dictionary at the previous level to give the dictionary at the next level. Random data projections are used to control the information flow to different layers. We also enforce a zero-tree constraint on the sparse representation weights so that the zero weights in the previous level will force the corresponding weights in the next level to be zero. At each stage we combine this zero-tree constraint with our new screening tests to reduce the size of Lasso problems that must be solved.

In detail, we use $l$ random projections $\mathbf{T}_k \in \mathbb{R}^{d_k \times p}$ ($1 \leq k \leq l$) to extract information incrementally from the data in $l$ stages. Each $\mathbf{T}_k$ has orthonormal rows and the rows of distinct $\mathbf{T}_k$ are orthogonal. At level $k$ we learn a dictionary $\mathbf{B}_k \in \mathbb{R}^{d_k \times m_k}$ and weights $\mathbf{W}_k \in \mathbb{R}^{m_k \times n}$ by solving a small sparse representation problem similar to (1):

$$\begin{aligned}
\min_{\mathbf{B}_k, \mathbf{W}_k} \quad & \frac{1}{2}\|\mathbf{T}_k\mathbf{X} - \mathbf{B}_k\mathbf{W}_k\|_F^2 + \lambda_k\|\mathbf{W}_k\|_1 \\
\text{s.t.} \quad & \|\mathbf{b}_i^{(k)}\|_2^2 \leq 1, \quad \forall i = 1, 2, \dots, m_k.
\end{aligned} \tag{9}$$

Here $\mathbf{b}_i^{(k)}$ is the $i^{th}$ column of matrix $\mathbf{B}_k$ and $m_k$ is assumed to be a multiple of $m_{k-1}$, so the number of codewords $m_k$ increases with $k$. We solve (9) for level $k = 1, 2, \dots, l$ sequentially.

An additional constraint is required to enforce a tree structure. Denote the $i^{th}$ element of the $j^{th}$ column of $\mathbf{W}_k$ by $\mathbf{w}_j^{(k)}(i)$ and organize the weights at level $k > 1$ in $m_{k-1}$ groups, one per level

$k - 1$ codeword. The $i^{th}$ group has $m_k/m_{k-1}$ weights: $\{\mathbf{w}_j^{(k)}(rm_{k-1} + i), 0 \leq r < m_k/m_{k-1}\}$, and has weight $\mathbf{w}_j^{(k-1)}(i)$ as its parent weight. To enforce a tree structure we require that a child weight is zero if its parent weight is zero. So for every level $k \geq 2$, data point $j$ ($1 \leq j \leq n$), group $i$ ($1 \leq i \leq m_{k-1}$), and weight $\mathbf{w}_j^{(k)}(rm_{k-1} + i)$ ($0 \leq r < m_k/m_{k-1}$), we enforce:

$$\mathbf{w}_j^{(k-1)}(i) = 0 \quad \Rightarrow \quad \mathbf{w}_j^{(k)}(rm_{k-1} + i) = 0. \tag{10}$$

This imposed tree structure is analogous to a "zero-tree" in EZW wavelet compression [22]. In addition, (10) means that the weights of the previous layer select a small subset of codewords to enter the Lasso optimization. When solving for $\mathbf{w}_j^k$, (10) reduces the number of codewords from $m_k$ to $(m_k/m_{k-1})\|\mathbf{w}_j^{(k-1)}\|_0$, a considerable reduction since $\mathbf{w}_j^{(k-1)}$ is sparse. Thus the screening rules in §2 and the imposed screening rule (10) work together to reduce the effective dictionary size.

The tree structure in the weights introduces a similar hierarchical tree structure in the dictionaries $\{\mathbf{B}_k\}_{k=1}^l$: the codewords $\{\mathbf{b}_{rm_{k-1}+i}^{(k)}, 0 \leq r < m_k/m_{k-1}\}$ are the children of codeword $\mathbf{b}_i^{(k-1)}$. This tree structure provides a heuristic way of updating $\mathbf{B}_k$. When $k > 1$, the $m_k$ codewords in layer $k$ are naturally divided into $m_{k-1}$ groups, so we can solve $\mathbf{B}_k$ by optimizing each group sequentially. This is similar to block coordinate descent. For $i = 1, 2, \ldots, m_{k-1}$, let $\mathbf{B}' = [\mathbf{b}_{rm_{k-1}+i}^{(k)}]_{r=0}^{m_k/m_{k-1}-1}$ denote the codewords in group $i$. Let $\mathbf{W}'$ be the submatrix of $\mathbf{W}$ containing only the $(rm_{k-1} + i)^{th}$ rows of $\mathbf{W}$, $r = 0, 1, \ldots, m_k/m_{k-1} - 1$. $\mathbf{W}'$ is the weight matrix for $\mathbf{B}'$. Denote the remaining codewords and weights by $\mathbf{B}''$ and $\mathbf{W}''$. For all $m_{k-1}$ groups in random order, we fix $\mathbf{B}''$ and update $\mathbf{B}'$ by solving (1) for data matrix $\mathbf{T}_k\mathbf{X} - \mathbf{B}''\mathbf{W}''$. This reduces the complexity from $O(m_k^q)$ to $O(m_k^q/m_{k-1}^{q-1})$ where $O(m^q)$ is the complexity of updating a dictionary with size $m$. Since $q \geq 3$, this offers big computational savings but might yield a suboptimal solution of (9).

After finalizing $\mathbf{W}_k$ and $\mathbf{B}_k$, we can solve an unconstrained QP to find $\mathbf{C}_k = \arg\min_{\mathbf{C}}\|\mathbf{X} - \mathbf{C}\mathbf{W}_k\|_F^2$. $\mathbf{C}_k$ is useful for visualization purposes; it represents the points on the original data manifold corresponding to $\mathbf{B}_k$.

In principle, our framework can use any orthogonal projection matrix $\mathbf{T}_k$. We choose random projections because they're simple and, more importantly, because they provide a mechanism to control the amount of information extracted at each layer. If all $\mathbf{T}_k$ are randomly generated independently of $\mathbf{X}$, then on average, the amount of information in $\mathbf{T}_k\mathbf{X}$ is proportional to $d_k$. This allows us to control the flow of information to each layer so that we avoid using all the information in one layer.

## 5   Experiments

We tested our framework on: **(a)** the COIL rotational image data set [23], **(b)** the MNIST digit classification data set [24], and **(c)** the extended Yale B face recognition data set [25] [26]. The basic sparse representation problem (1) is solved using the toolbox provided in [12] to iteratively optimize $\mathbf{B}$ and $\mathbf{W}$ until an iteration results in a loss function reduction of less than $0.01\%$.

**COIL Rotational Image Data Set:** This is intended as a small scale illustration of our framework. We use the 72, 128x128 color images of object No. 80 rotating around a circle in 15 degree-increments (18 images shown in Fig.2(a)). We ran the traditional sparse representation algorithm to compare the three screening tests in §2. The dictionary size is $m = 16$ and we vary $\lambda$. As shown in Fig.2(c), ST3 discards a larger fraction of codewords than ST2 and ST2 discards a larger fraction than ST1/SAFE. We ran the same algorithms on 200 random data projections and the results are almost identical. The average $\lambda_{\max}$ for these two situations is 0.98.

Next we test our hierarchical framework using two layers. We set $(d_2, m_2) = (200, 16)$ so that the second layer solves a problem of the same scale as in the previous paragraph. We demonstrate how the result of the first layer, with $(d_1, m_1, \lambda_1) = (100, 4, 0.5)$, helps the second layer discard more codewords when the tree constraint (10) is imposed. Fig.2(b) illustrates this constraint: the 16 second layer codewords are organized in 4 groups of 4 (only 2 groups shown). The weight on any codeword in a group has to be zero if the parent codeword in the first layer has weight zero. This imposed constraint discards many more codewords in the screening stage than any of the three tests in §2. (Fig.2(d)). Finally, the illustrated codewords and weights in Fig.2(b) are the actual values in

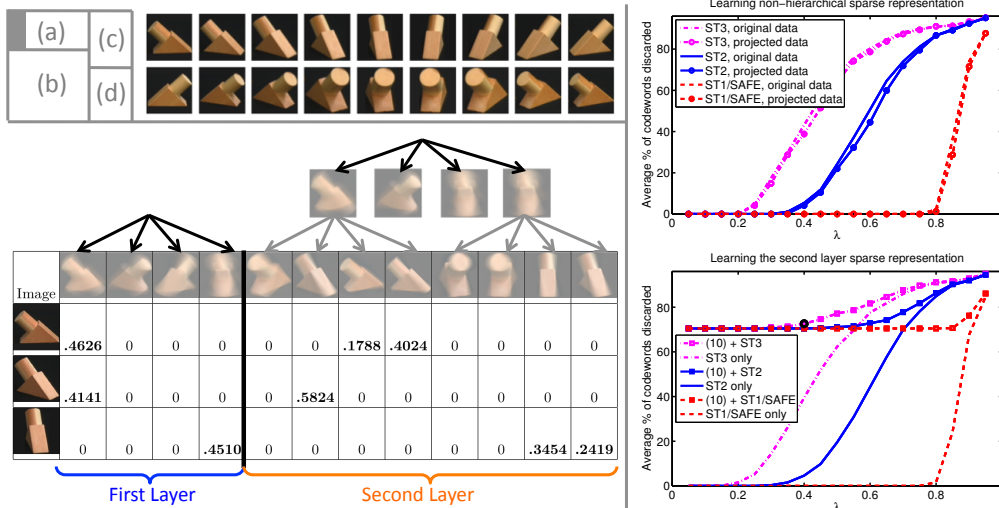

Figure 2: **(a):** Example images of the data set. **(b):** Illustration of a two layer hierarchical sparse representation. **(c):** Comparison of the three screening tests for sparse representation. **(d):** Screening performance in the second layer of our hierarchical framework using combinations of screening criteria. The imposed constraint (10) helps to discard significantly more codewords when $\lambda$ is small.

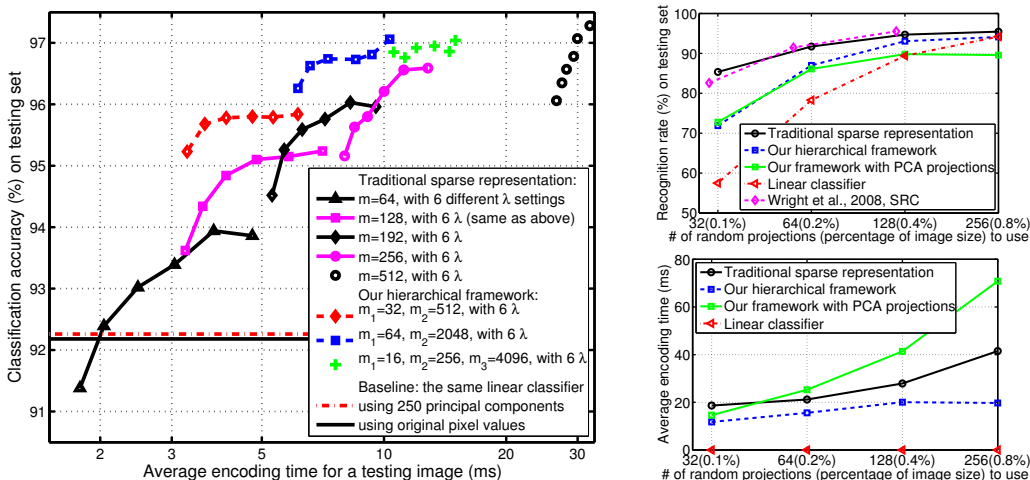

Figure 3: **Left:** MNIST: The tradeoff between classification accuracy and average encoding time for various sparse representation methods. Our hierarchical framework yields better performance in less time. The average encoding time doesn't apply to baseline methods. The performance of traditional sparse representation is consistent with [9]. **Right:** Face Recognition: The recognition rate (top) and average encoding time (bottom) for various methods. Traditional sparse representation has the best accuracy and is very close to a similar method SRC in [8] (SRC's recognition rate is cited from [8] but data on encoding time is not available). Our hierarchical framework achieves a good tradeoff between the accuracy and speed. Using PCA projections in our framework yields worse performance since these projections do not spread information across the layers.

$\mathbf{C}_2$ and $\mathbf{W}_2$ when $\lambda_2 = 0.4$ (the marked point in Fig.2(d)). The sparse representation gives a multi-resolution representation of the rotational pattern: the first layer encodes rough orientation and the second layer refines it.

The next two experiments evaluate the performance of sparse representation by **(1)** the accuracy of a classification task using the columns in $\mathbf{W}$ (or in $[\mathbf{W}_1^T, \mathbf{W}_2^T, \ldots, \mathbf{W}_l^T]^T$ for our framework) as features, and **(2)** the average encoding time required to obtain these weights for a testing data point. This time is highly correlated with the total time needed for iterative dictionary learning. We used linear SVM (liblinear [27]) with parameters tuned by 10-fold cross-validations on the training set.

**MNIST Digit Classification:** This data set contains 70,000 28x28 hand written digit images (60,000 training, 10,000 testing). We ran the traditional sparse representation algorithm for dictionary size $m \in \{64, 128, 192, 256\}$ and $\lambda \in \Lambda = \{0.06, 0.08, 0.11, 0.16, 0.23, 0.32\}$. In Fig.3 left panel, each curve contains settings with the same $m$ but with different $\lambda$. Points to the right correspond to smaller $\lambda$ values (less sparse solutions and more difficult computation). There is a tradeoff between speed (x-axis) and classification performance (y-axis). To see where our framework stands, we tested the following settings: **(a)** 2 layers with $(d_1, d_2) = (200, 500), (m_1, m_2) = (32, 512)$, $\lambda_1 = 0.23$ and $\lambda_2 \in \Lambda$, **(b)** $(m_1, m_2) = (64, 2048)$ and everything else in (a) unchanged, **(c)** 3 layers with $(d_1, d_2, d_3) = (100, 200, 400), (m_1, m_2, m_3) = (16, 256, 4096), (\lambda_1, \lambda_2) = (0.16, 0.11)$ and $\lambda_3 \in \Lambda$. The plot shows that compared to the traditional sparse representation, our hierarchical framework achieves roughly a $1\%$ accuracy improvement given the same encoding time and a roughly 2X speedup given the same accuracy. Using 3 layers also offers competitive performance but doesn't outperform the 2 layer setting.

**Face Recognition:** For each of 38 subjects we used 64 cropped frontal face views under differing lighting conditions, randomly divided into 32 training and 32 testing images. This set-up mirrors that in [8]. In this experiment we start with the random projected data ($p \in \{32, 64, 128, 256\}$ random projections of the original 192x128 data) and use this data as follows: **(a)** learn a traditional non-hierarchical sparse representation, **(b)** our framework, i.e., sample the data in two stages using orthogonal random projections and learn a 2 layer hierarchical sparse representation, **(c)** use PCA projections to replace random projections in (b), **(d)** directly apply a linear classifier without first learning a sparse representation. For (a) we used $m = 1024$, $\lambda = 0.030$ for $p = 32, 64$ and $\lambda = 0.029$ for $p = 128, 256$ (tuned to yield the same average sparsity for different $p$). For (b) we used $(m_1, m_2) = (32, 1024), (d_1, d_2) = (\frac{3}{8}p, \frac{5}{8}p)$, $\lambda_1 = 0.02$ and $\lambda_2$ the same as $\lambda$ in (a). For (c) we used the same setting in (b) except random projection matrices $\mathbf{T}_1, \mathbf{T}_2$ in our framework are now set to the PCA projection matrices (calculate SVD $\mathbf{X} = \mathbf{U}\mathbf{S}\mathbf{V}^T$ with singular values in descending order, then use the first $d_1$ columns of $\mathbf{U}$ as the rows in $\mathbf{T}_1$ and the next $d_2$ columns of $\mathbf{U}$ as the rows in $\mathbf{T}_2$). The results in Fig.3 right panel suggest that our framework strikes a good balance between speed and accuracy. The PCA variant of our framework has worse performance because the first $\frac{3}{8}p$ projections contain too much information, leaving the second layer too little information (which also drags down the speed for lack of sparsity and structure). This reinforces our argument at the end of §4 about the advantage of random projections. The fact that a linear SVM performs well given enough random projections suggests this data set does not have a strong nonlinear structure.

Finally, at any iteration, the average $\lambda_{\max}$ for all data points ranges from $0.76$ to $0.91$ in all settings in the MNIST experiment and ranges from $0.82$ to nearly 1 in the face recognition experiment (except for the second layer in the PCA variant, in which average $\lambda_{\max}$ can be as low as $0.54$). As expected, $\lambda_{\max}$ is large, a situation that favors our new screening tests (ST2, ST3).

# 6    Conclusion

Our theoretical results and algorithmic framework effectively make headway on the computational challenge of learning sparse representations on large size dictionaries for high dimensional data The new screening tests greatly reduce the size of the lasso problems to be solved and the tests are proven, both theoretically and empirically, to be much more effective than the existing ST1/SAFE test. We have shown that under certain conditions, random projection preserves the scale indifference (SI) property with high probability, thus providing an opportunity to learn informative sparse representations with data fewer dimensions. Finally, the new hierarchical dictionary learning framework employs random data projections to control the flow of information to the layers, screening to eliminate unnecessary codewords, and a tree constraint to select a small number of candidate codewords based on the weights leant in the previous stage. By doing so, it can deal with large $m$ and $p$ simultaneously. The new framework exhibited impressive performance on the tested data sets, achieving equivalent classification accuracy with less computation time.

**Acknowledgements**

This research was partially supported by the NSF grant CCF-1116208. Zhen James Xiang thanks Princeton University for support through the Charlotte Elizabeth Procter honorific fellowship.

# References

[1] M. Elad. *Sparse and Redundant Representations: From Theory to Applications in Signal and Image Processing*. Springer, 2010.

[2] G. Cao and C.A. Bouman. Covariance estimation for high dimensional data vectors using the sparse matrix transform. In *Advances in Neural Information Processing Systems*, 2008.

[3] A.B. Lee, B. Nadler, and L. Wasserman. Treelets An adaptive multi-scale basis for sparse unordered data. *The Annals of Applied Statistics*, 2(2):435–471, 2008.

[4] M. Gavish, B. Nadler, and R.R. Coifman. Multiscale wavelets on trees, graphs and high dimensional data: Theory and applications to semi supervised learning. In *International Conference on Machine Learning*, 2010.

[5] M. Belkin and P. Niyogi. Using manifold stucture for partially labeled classification. In *Advances in Neural Information Processing Systems*, pages 953–960, 2003.

[6] S.T. Roweis and L.K. Saul. Nonlinear dimensionality reduction by locally linear embedding. *Science*, 290(5500):2323, 2000.

[7] B.A. Olshausen and D.J. Field. Sparse coding with an overcomplete basis set: A strategy employed by V1? *Vision research*, 37(23):3311–3325, 1997.

[8] J. Wright, A.Y. Yang, A. Ganesh, S.S. Sastry, and Y. Ma. Robust face recognition via sparse representation. *IEEE Transactions on Pattern Analysis and Machine Intelligence*, 31(2):210–227, 2008.

[9] K. Yu, T. Zhang, and Y. Gong. Nonlinear learning using local coordinate coding. In *Advances in Neural Information Processing Systems*, volume 3, 2009.

[10] B. Efron, T. Hastie, I. Johnstone, and R. Tibshirani. Least angle regression. *Annals of Statistics*, pages 407–451, 2004.

[11] J. Mairal, F. Bach, J. Ponce, and G. Sapiro. Online learning for matrix factorization and sparse coding. *The Journal of Machine Learning Research*, 11:19–60, 2010.

[12] H. Lee, A. Battle, R. Raina, and A.Y. Ng. Efficient sparse coding algorithms. In *Advances in Neural Information Processing Systems*, volume 19, page 801, 2007.

[13] L.E. Ghaoui, V. Viallon, and T. Rabbani. Safe feature elimination in sparse supervised learning. *Arxiv preprint arXiv:1009.3515*, 2010.

[14] R. Tibshirani, J. Bien, J. Friedman, T. Hastie, N. Simon, J. Taylor, and R.J. Tibshirani. Strong rules for discarding predictors in lasso-type problems. *Arxiv preprint arXiv:1011.2234*, 2010.

[15] J. Wright, Y. Ma, J. Mairal, G. Sapiro, T. Huang, and S. Yan. Sparse representation for computer vision and pattern recognition. *Proceedings of the IEEE*, 98(6):1031–1044, 2010.

[16] R.G. Baraniuk and M.B. Wakin. Random projections of smooth manifolds. *Foundations of Computational Mathematics*, 9(1):51–77, 2007.

[17] Y. Lin, T. Zhang, S. Zhu, and K. Yu. Deep coding network. In *Advances in Neural Information Processing Systems*, 2010.

[18] G.E. Hinton, S. Osindero, and Y.W. Teh. A fast learning algorithm for deep belief nets. *Neural Computation*, 18(7):1527–1554, 2006.

[19] R. Jenatton, J. Mairal, G. Obozinski, and F. Bach. Proximal methods for sparse hierarchical dictionary learning. In *International Conference on Machine Learning*, 2010.

[20] M.B. Wakin, D.L. Donoho, H. Choi, and R.G. Baraniuk. Highresolution navigation on non-differentiable image manifolds. In *IEEE International Conference on Acoustics, Speech and Signal Processing*, volume 5, pages 1073–1076, 2005.

[21] M.F. Duarte, M.A. Davenport, M.B. Wakin, JN Laska, D. Takhar, K.F. Kelly, and RG Baraniuk. Multiscale random projections for compressive classification. In *IEEE International Conference on Image Processing*, volume 6, 2007.

[22] J.M. Shapiro. Embedded image coding using zerotrees of wavelet coefficients. *IEEE Transactions on Signal Processing*, 41(12):3445–3462, 2002.

[23] S.A. Nene, S.K. Nayar, and H. Murase. Columbia object image library (coil-100). *Techn. Rep. No. CUCS-006-96, dept. Comp. Science, Columbia University*, 1996.

[24] Y. Lecun, L. Bottou, Y. Bengio, and P. Haffner. Gradient-based learning applied to document recognition. *Proceedings of the IEEE*, 86(11):2278 –2324, nov 1998.

[25] A.S. Georghiades, P.N. Belhumeur, and D.J. Kriegman. From few to many: Illumination cone models for face recognition under variable lighting and pose. *IEEE Transactions on Pattern Analysis and Machine Intelligence*, 23(6):643–660, 2002.

[26] K.C. Lee, J. Ho, and D.J. Kriegman. Acquiring linear subspaces for face recognition under variable lighting. *IEEE Transactions on Pattern Analysis and Machine Intelligence*, pages 684–698, 2005.

[27] R.E. Fan, K.W. Chang, C.J. Hsieh, X.R. Wang, and C.J. Lin. LIBLINEAR: A library for large linear classification. *The Journal of Machine Learning Research*, 9:1871–1874, 2008.

